# Contextual Gaussian Process Bandit Optimization

**Andreas Krause**　　　　　　**Cheng Soon Ong**
Department of Computer Science, ETH Zurich,
8092 Zurich, Switzerland
krausea@ethz.ch　　　chengsoon.ong@inf.ethz.ch

## Abstract

How should we design experiments to maximize performance of a complex system, taking into account uncontrollable environmental conditions? How should we select relevant documents (ads) to display, given information about the user? These tasks can be formalized as contextual bandit problems, where at each round, we receive context (about the experimental conditions, the query), and have to choose an action (parameters, documents). The key challenge is to trade off exploration by gathering data for estimating the mean payoff function over the context-action space, and to exploit by choosing an action deemed optimal based on the gathered data. We model the payoff function as a sample from a Gaussian process defined over the joint context-action space, and develop CGP-UCB, an intuitive upper-confidence style algorithm. We show that by mixing and matching kernels for contexts and actions, CGP-UCB can handle a variety of practical applications. We further provide generic tools for deriving regret bounds when using such composite kernel functions. Lastly, we evaluate our algorithm on two case studies, in the context of automated vaccine design and sensor management. We show that context-sensitive optimization outperforms no or naive use of context.

## 1   Introduction

Consider the problem of learning to optimize a complex system subject to varying environmental conditions. Or learning to retrieve relevant documents (ads), given context about the user. Or learning to solve a sequence of related optimization and search tasks, by taking into account experience with tasks solved previously. All these problems can be phrased as a *contextual bandit problem* (*c.f.*, [1, 2], we review related work in Section 7), where in each round, we receive context (about the experimental conditions, the query, or the task), and have to choose an action (system parameters, document to retrieve). We then receive noisy feedback about the obtained payoff. The key challenge is to trade off exploration by gathering data for estimating the mean payoff function over the context-action space, and to exploit by choosing an action deemed optimal based on the gathered data.

Without making any assumptions about the class of payoff functions under consideration, we cannot expect to do well. A natural approach is to choose a regularizer, encoding assumptions about smoothness of the payoff function. In this paper, we take a nonparametric approach, and model the payoff function as a sample from a Gaussian process defined over the joint context-action space (or having low norm in the associated RKHS). This approach allows us to estimate the predictive uncertainty in the payoff function estimated from previous experiments, guiding the tradeoff between exploration and exploitation. In the context-free case, this problem is studied by [3], who analyze GP-UCB, an upper-confidence bound-based sampling algorithm that makes use of the predictive uncertainty to trade exploration and exploitation. In this paper, we develop CGP-UCB, a natural generalization of GP-UCB, which takes context information into account. By constructing a composite kernel function for the regularizer from kernels defined over the action and context spaces (e.g., a linear kernel on the actions, and Gaussian kernel on the contexts), we can capture several natural contextual bandit problem formulations. We prove that CGP-UCB incurs

sublinear *contextual* regret (i.e., prove that it competes with the optimal mapping from context to actions) for a large class of composite kernel functions constructed in this manner. Lastly, we evaluate our algorithm on two real-world case studies in the context of automated vaccine design, and management of sensor networks. We show that in both these problems, properly taking into account contextual information outperforms ignoring or naively using context.

In summary, as our main contributions we

- develop an efficient algorithm, CGP-UCB, for the contextual GP bandit problem;
- show that by flexibly combining kernels over contexts and actions, CGP-UCB can be applied to a variety of applications;
- provide a generic approach for deriving regret bounds for composite kernel functions;
- evaluate CGP-UCB on two case studies, related to automated vaccine design and sensor management.

## 2 Modeling Contextual Bandits with Gaussian Processes

We consider playing a game for a sequence of $T$ (not necessarily known a priori) rounds. In each round, we receive a *context* $\mathbf{z}_t \in Z$ from a (not necessarily finite) set $Z$ of contexts, and have to choose an action $\mathbf{s}_t \in S$ from a (not necessarily finite) set $S$ of actions. We then receive a payoff $y_t = f(\mathbf{s}_t, \mathbf{z}_t) + \epsilon_t$, where $f : S \times Z \to \mathbb{R}$ is an (unknown) function, and $\epsilon_t$ is zero mean random noise (independent across the rounds). The addition of (externally chosen) contextual information captures a critical component in many applications, and generalizes the k-armed bandit setting.

Since $f$ is unknown, we will not generally be able to choose the optimal action, and thus incur regret $r_t = \sup_{\mathbf{s}' \in S} f(\mathbf{s}', \mathbf{z}_t) - f(\mathbf{s}_t, \mathbf{z}_t)$. After $T$ rounds, our cumulative regret is $R_T = \sum_{t=1}^T r_t$. The context-specific best action is a more demanding benchmark than the best action used in the (context-free) definition of regret. Our goal will be to develop an algorithm which achieves sublinear contextual regret, i.e., $R_T / T \to 0$ for $T \to \infty$. Note that achieving sublinear contextual regret requires learning (and competing with) the optimal mapping from contexts to actions.

Regularity assumptions are required, since without any there could be a single action $\mathbf{s}^* \in S$ that obtains payoff of 1, and all other actions obtain payoff 0. With infinite action sets, no algorithm will be able to identify $\mathbf{s}^*$ in finite time. In this paper, we assume that the function $f : S \times Z \to \mathbb{R}$ is a sample from a known *Gaussian process* (GP) distribution[1]. A *Gaussian process* is a collection of dependent random variables, one for each $\mathbf{x} \in X$, such that every finite marginal distribution is a multivariate Gaussian (while ensuring overall consistency) [4]. Here we use $X = S \times Z$ to refer to the set of all action-context pairs. A $GP(\mu, k)$ is fully specified by its mean function $\mu : X \to \mathbb{R}$, $\mu(\mathbf{x}) = \mathbb{E}[f(\mathbf{x})]$ and covariance (or kernel) function $k : X \times X \to \mathbb{R}$, $k(\mathbf{x}, \mathbf{x}') = \mathbb{E}[(f(\mathbf{x}) - \mu(\mathbf{x}))(f(\mathbf{x}') - \mu(\mathbf{x}'))]$. Without loss of generality [4], we assume that $\mu \equiv 0$. We further assume bounded variance by restricting $k(\mathbf{x}, \mathbf{x}) \leq 1$, for all $\mathbf{x} \in X$. The covariance function $k$ encodes smoothness properties of sample functions $f$ drawn from the GP. Since the random variables are action-context pairs, often there is a natural decomposition of the covariance function $k$ into the corresponding covariance functions on actions and contexts (Section 5).

A major computational benefit of working with GPs is the fact that posterior inference can be performed in closed form. Suppose we have collected observations $\mathbf{y}_T = [y_1 \ldots y_T]^T$ at inputs $A_T = \{\mathbf{x}_1, \ldots, \mathbf{x}_T\}$, $y_t = f(\mathbf{x}_t) + \epsilon_t$ with i.i.d. Gaussian noise $\epsilon_t \sim N(0, \sigma^2)$, the posterior distribution over $f$ is a GP with mean $\mu_T(\mathbf{x})$, covariance $k_T(\mathbf{x}, \mathbf{x}')$ and variance $\sigma_T^2(\mathbf{x})$, with parameters estimated as

$$\mu_T(\mathbf{x}) = \boldsymbol{k}_T(\boldsymbol{x})^T (\boldsymbol{K}_T + \sigma^2 \boldsymbol{I})^{-1} \boldsymbol{y}_T,$$
$$k_T(\mathbf{x}, \mathbf{x}') = k(\mathbf{x}, \mathbf{x}') - \boldsymbol{k}_T(\mathbf{x})^T (\boldsymbol{K}_T + \sigma^2 \boldsymbol{I})^{-1} \boldsymbol{k}_T(\mathbf{x}'),$$
$$\sigma_T^2(\mathbf{x}) = k_T(\mathbf{x}, \mathbf{x}),$$

where $\boldsymbol{k}_T(\mathbf{x}) = [k(\mathbf{x}_1, \mathbf{x}) \ldots k(\mathbf{x}_T, \mathbf{x})]^T$ and $\boldsymbol{K}_T$ is the (positive semi-definite) kernel matrix $[k(\mathbf{x}, \mathbf{x}')]_{\mathbf{x}, \mathbf{x}' \in A_T}$. The choice of the kernel function turns out to be crucial in regularizing the function class to achieve sublinear regret (Section 4).

## 3 The Contextual Upper Confidence Bound Algorithm

In the *context-free* case $Z = \emptyset$, the problem of trading off exploration and exploitation with payoff functions sampled from a Gaussian process is studied by [3]. They show that a simple upper confidence bound algorithm, GP-UCB (Equation 1), achieves sublinear regret. At round $t$, GP-UCB picks action $\mathbf{s}_t = \mathbf{x}_t$ such that

$$\mathbf{s}_t = \underset{\mathbf{s} \in S}{\operatorname{argmax}} \, \mu_{t-1}(\mathbf{s}) + \beta_t^{1/2} \sigma_{t-1}(\mathbf{s}), \qquad (1)$$

where $\beta_t$ are appropriate constants. Here $\mu_{t-1}(\cdot)$ and $\sigma_{t-1}(\cdot)$ are the posterior mean and standard deviation conditioned on the observations $(\mathbf{s}_1, y_1), \ldots, (\mathbf{s}_{t-1}, y_{t-1})$. This GP-UCB objective naturally trades off exploration (picking actions with uncertain outcomes, i.e., large $\sigma_{t-1}(\mathbf{s})$), and exploitation (picking actions expected to do well, i.e., having large $\mu_{t-1}(\mathbf{s})$).

We propose a natural generalization of GP-UCB, which incorporates contextual information

$$\mathbf{s}_t = \underset{\mathbf{s} \in S}{\operatorname{argmax}} \, \mu_{t-1}(\mathbf{s}, \mathbf{z}_t) + \beta_t^{1/2} \sigma_{t-1}(\mathbf{s}, \mathbf{z}_t), \qquad (2)$$

where $\mu_{t-1}(\cdot)$ and $\sigma_{t-1}(\cdot)$ are the posterior mean and standard deviation of the GP over the joint set $X = S \times Z$ conditioned on the observations $(\mathbf{s}_1, \mathbf{z}_1, y_1), \ldots, (\mathbf{s}_{t-1}, \mathbf{z}_{t-1}, y_{t-1})$. Thus, when presented with context $\mathbf{z}_t$, this algorithm uses posterior inference to predict mean and variance for each possible decision $\mathbf{s}$, conditioned on all past observations (involving both the chosen actions, the observed contexts as well as the noisy payoffs). We call the greedy algorithm implementing rule 2 the *contextual Gaussian process UCB algorithm* (CGP-UCB). As we will show in Section 5, this algorithm allows to incorporate various assumptions about the dependencies of the payoff function on the chosen actions and observed contexts. It also allows us to generalize several approaches proposed in the literature [3, 5, 6]. In the following, we will prove that in many practical applications, CGP-UCB attains sublinear contextual regret (i.e., is able to compete with the optimal mapping from contexts to actions).

## 4 Bounds on the Contextual Regret

Bounding the *contextual regret* of CGP-UCB is a challenging problem, since the regret is measured with respect to the best action for each context. Intuitively, the amount of regret we incur should depend on how quickly we can gather information about the payoff function, which now jointly depends on context and actions. In the following, we show that the contextual regret of CGP-UCB is bounded by an intuitive information-theoretic quantity, which quantifies the mutual information between the observed context-action pairs and the estimated payoff function $f$.

We start by reviewing the special case of [3] where no context information is provided. It is shown that in this context-free case, the regret $R_T$ of the GP-UCB algorithm can be bounded as $\mathcal{O}^*(\sqrt{T\gamma_T})$, where $\gamma_T$ is defined as:

$$\gamma_T := \max_{A \subset S : |A| = T} \mathrm{I}(\boldsymbol{y}_A; f),$$

where $\mathrm{I}(\boldsymbol{y}_A; f) = \mathrm{H}(\boldsymbol{y}_A) - \mathrm{H}(\boldsymbol{y}_A | f)$ quantifies the reduction in uncertainty (measured in terms of differential Shannon entropy [7]) about $f$ achieved by revealing $\boldsymbol{y}_A$. In the multivariate Gaussian case, the entropy can be computed in closed form: $\mathrm{H}(N(\boldsymbol{\mu}, \boldsymbol{\Sigma})) = \frac{1}{2} \log |2\pi e \boldsymbol{\Sigma}|$, so that $\mathrm{I}(\boldsymbol{y}_A; f) = \frac{1}{2} \log |\boldsymbol{I} + \sigma^{-2} \boldsymbol{K}_A|$, where $\boldsymbol{K}_A = [k(\mathbf{s}, \mathbf{s}')]_{\mathbf{s}, \mathbf{s}' \in A}$ is the Gram matrix of $k$ evaluated on set $A \subseteq S$.

For the contextual case, our regret bound comes also in terms of the quantity $\gamma_T$, redefined so that the information gain $\mathrm{I}(\boldsymbol{y}_A; f)$ now depends on the observations $\boldsymbol{y}_A = [y(\mathbf{x})]_{\mathbf{x} \in A}$ of the joint context-action pairs $\mathbf{x} = (\mathbf{s}, \mathbf{z})$, and $f : S \times Z \to \mathbb{R}$ is the payoff function over the context-action space. Consequently, the kernel matrix $\boldsymbol{K}_A = [k(\mathbf{x}, \mathbf{x}')]_{\mathbf{x}, \mathbf{x}' \in A}$ is defined over context-action pairs. Using this notion of information gain $\gamma_T$, we lift the results of [3] to the much more general contextual bandit setting, shedding further light on the connection between bandit optimization and information gain. In Section 5, we show how to bound $\gamma_T$ for composite kernels, combining possibly different assumptions about the regularity of $f$ in the action space $S$ and context space $Z$.

We consider the same three settings as analyzed in [3]. Note that none of the results subsume each other, and so all cases may be of use. For the first two settings, we assume a known GP prior and (1) a finite $X$ and (2) infinite $X$ with mild assumptions about $k$. A third (and perhaps more "agnostic") way to express assumptions about $f$ is to require that $f$ has low "complexity" as quantified in terms of the Reproducing Kernel Hilbert Space (RKHS, [8]) norm associated with kernel $k$.

**Theorem 1** *Let $\delta \in (0, 1)$. Suppose one of the following assumptions holds*

1. *$X$ is finite, $f$ is sampled from a known GP prior with known noise variance $\sigma^2$, and $\beta_t = 2\log(|X|t^2\pi^2/6\delta)$*

2. *$X \subseteq [0, r]^d$ is compact and convex, $d \in \mathbb{N}$, $r > 0$. Suppose $f$ is sampled from a known GP prior with known noise variance $\sigma^2$, and that $k(\mathbf{x}, \mathbf{x}')$ satisfies the following high probability bound on the derivatives of GP sample paths $f$: for some constants $a, b > 0$,*

$$\Pr\left\{\sup_{\boldsymbol{x}\in X} |\partial f/\partial x_j| > L\right\} \le ae^{-(L/b)^2}, \quad j = 1, \ldots, d.$$

   *Choose $\beta_t = 2\log(t^2 2\pi^2/(3\delta)) + 2d\log\left(t^2 dbr\sqrt{\log(4da/\delta)}\right)$.*

3. *$X$ is arbitrary; $||f||_k \le B$. The noise variables $\epsilon_t$ form an* arbitrary *martingale difference sequence (meaning that $\mathbb{E}[\varepsilon_t \,|\, \varepsilon_1, \ldots, \varepsilon_{t-1}] = 0$ for all $t \in \mathbb{N}$), uniformly bounded by $\sigma$. Further define $\beta_t = 2B^2 + 300\gamma_t \ln^3(t/\delta)$.*

*Then the* contextual regret *of CGP-UCB is bounded by $\mathcal{O}^*(\sqrt{T\gamma_T\beta_T})$ w.h.p. Precisely,*

$$\Pr\left\{R_T \le \sqrt{C_1 T \beta_T \gamma_T} + 2 \quad \forall T \ge 1\right\} \ge 1 - \delta.$$

*where $C_1 = 8/\log(1 + \sigma^{-2})$.*

Theorem 1 (proof given in the supplemental material) shows that, in case (1) and (2), with high probability over samples from the GP, the cumulative contextual regret is bounded in terms of the maximum information gain with respect to the GP defined over $S \times Z$. In case of assumption (3), a regret bound is obtained in a more agnostic setting, where no prior on $f$ is assumed, and much weaker assumptions are made about the noise process. Note that case (3) requires a bound $B$ on $||f||_k$. If no such bound is available, standard guess-and-doubling arguments can be used.

## 5 Applications of CGP-UCB

By choosing different kernel functions $k : X \times X \to \mathbb{R}$, the CGP-UCB algorithm can be applied to a variety of different applications. A natural approach is to start with kernel functions $k_Z : Z \times Z \to \mathbb{R}$ and $k_S : S \times S \to \mathbb{R}$ on the space of contexts and actions, and use them to derive the kernel on the product space.

### 5.1 Constructing Composite Kernels

One possibility is to consider a product kernel $k = k_S \otimes k_Z$, by setting $(k_S \otimes k_Z)((\mathbf{s}, \mathbf{z}), (\mathbf{s}', \mathbf{z}')) = k_Z(\mathbf{z}, \mathbf{z}')k_S(\mathbf{s}, \mathbf{s}')$. The intuition behind this product kernel is a conjunction of the notions of similarities induced by the kernels over context and action spaces: Two context-action pairs are similar (large correlation) if the contexts are similar and actions are similar (Figure 1(a)). Note that many kernel functions used in practice are already in product form. For example, if $k_Z$ and $k_S$ are squared exponential kernels (or Matérn kernels with smoothness parameters $\nu$), then the product $k = k_Z \otimes k_S$ is a squared exponential kernel (or Matérn kernels with smoothness parameters $\nu$). Similarly, if $k_S$

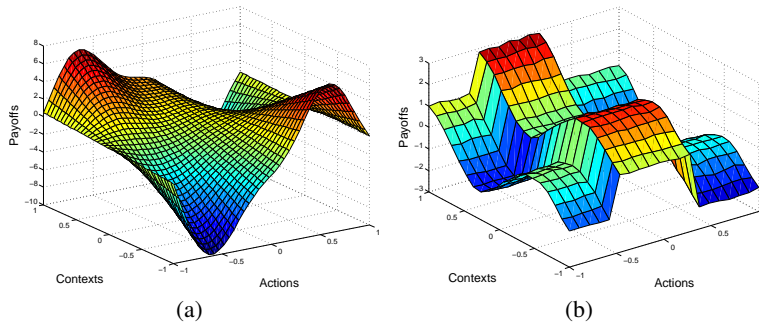

(a)                    (b)

Figure 1: Illustrations of composite kernel functions that can be incorporated into CGP-UCB. (a) Product of squared exponential kernel and linear kernel; (b) additive combination of a payoff function that smoothly depends on context, and exhibits clusters of actions. In general, context and action spaces are higher dimensional.

and $k_Z$ have finite rank $m_S$ and $m_Z$ (i.e., all kernel matrices over finite sets have rank at most $m_S$ and $m_Z$ respectively), then $k_S \otimes k_Z$ has finite rank $m_S m_Z$. However, other kernel functions can be naturally combined as well.

An alternative is to consider the additive combination $(k_S \oplus k_Z)((\mathbf{s}, \mathbf{z}), (\mathbf{s}', \mathbf{z}')) = k_Z(\mathbf{z}, \mathbf{z}') + k_S(\mathbf{s}, \mathbf{s}')$ which is positive definite as well. The intuition behind this construction is that a GP with additive kernel can be understood as a generative model, which first samples a function $f_S(\mathbf{s}, \mathbf{z})$ that is constant along $\mathbf{z}$, and varies along $\mathbf{s}$ with regularity as expressed by $k_{\mathbf{s}}$; it then samples a function $f_{\mathbf{z}}(\mathbf{s}, \mathbf{z})$, which varies along $\mathbf{z}$ and is constant along $\mathbf{s}$; then $f = f_{\mathbf{s}} + f_{\mathbf{z}}$. Thus, the $f_{\mathbf{z}}$ component models overall trends according to the context (e.g., encoding assumptions about similarity within clusters of contexts), and the $f_S$ models action-specific deviation from this trend (Figure 1(b)). In Section 5.3, we provide examples of applications that can be captured in this framework.

## 5.2 Bounding the Information Gain for Composite Kernels.

Since the key quantity governing the regret is the information gain $\gamma_T$, we would like to find a convenient way of bounding $\gamma_T$ for composite kernels ($k_S \otimes k_Z$ and $k_S \oplus k_Z$), plugging in different regularity assumptions for the contexts (via $k_Z$) and actions (via $k_S$). More formally, let us define

$$\gamma(T; k; V) = \max_{A \subseteq V, |A| \leq T} \frac{1}{2} \log \left| \boldsymbol{I} + \sigma^{-2} [k(\mathbf{v}, \mathbf{v}')]_{\mathbf{v}, \mathbf{v}' \in A} \right|,$$

which quantifies the maximum possible information gain achievable by sampling $T$ points in a GP defined over set $V$ with kernel function $k$. In [3, Theorem 5], bounds on $\gamma(T; k; V)$ were derived for common kernel functions including the linear ($\gamma(T; k; V) = \mathcal{O}(d \log T)$ for $d$-dimensions), the squared exponential ($\gamma(T; k; V) = \mathcal{O}((\log T)^{d+1})$) and Matérn kernels ($\gamma(T; k; V) = \mathcal{O}(T^{d(d+1)/(2\nu+d(d+1))} \log T)$ for smoothness parameter $\nu$).

In the following, we show how $\gamma(T; k; V)$ can be bounded for composite kernels of the form $k_S \otimes k_Z$ and $k_S \oplus k_Z$, dependent on $\gamma(T; k_S; S)$ and $\gamma(T; k_Z; Z)$.

**Theorem 2** *Let $k_Z$ be a kernel function on $Z$ with rank at most $d$ (i.e., all Gram matrices over arbitrary finite sets of points $A \subseteq Z$ have rank at most $d$). Then*

$$\gamma(T; k_S \otimes k_Z; X) \leq d\gamma(T; k_S; S) + d \log T.$$

The assumptions of Theorem 2 are satisfied, for example, if $|Z| < \infty$ and $\mathrm{rk}\,\mathbf{K}_Z = d$, or if $k_Z$ is a $d$-dimensional linear kernel on $Z \subseteq \mathbb{R}^d$. Theorem 2 also holds with the roles of $k_Z$ and $k_S$ reversed.

**Theorem 3** *Let $k_S$ and $k_Z$ be kernel functions on $S$ and $Z$ respectively. Then for the additive combination $k = k_S \oplus k_Z$ defined on $X$ it holds that*

$$\gamma(T; k_S \oplus k_Z; X) \leq \gamma(T; k_S; S) + \gamma(T; k_Z; Z) + 2 \log T.$$

Proofs of Theorems 2 and 3 are given in the supplemental material. By combining the results above with the information gain bounds of [3], we can immediately obtain that, e.g., $\gamma_T$ for the product of a $d_1$ dimensional linear kernel and a $d_2$ dimensional Gaussian kernel is $\mathcal{O}(d_1 (\log T)^{d_2+1})$.

## 5.3 Example applications.

We now illustrate the generality of the CGP-UCB approach, by fleshing out four possible applications. In Section 6, we experimentally evaluate CGP-UCB on two of these applications.

**Online advertising and news recommendation.** Suppose an online service would like to display query-specific ads. This is the textbook contextual bandit problem [9]. There are $|S| = m$ different ads to select from, and each round we receive, for each ad $\mathbf{s} \in S$, a feature vector $\mathbf{z_s}$. Thus, the complete context is $\mathbf{z} = [\mathbf{z}_1, \ldots, \mathbf{z}_m]$. [9] model the expected payoff for each action as a (unknown) linear function $\mu(\mathbf{s}, \mathbf{z}) = \mathbf{z_s}^T \theta_{\mathbf{s}}^*$. Hereby, $\theta_{\mathbf{s}}^*$ models the dependence of action $\mathbf{s}$ on the context $\mathbf{z}$. Besides online advertising, a similar model has been proposed and experimentally studied by [6] for the problem of contextual news recommendation (see Section 7 for a discussion). Both these problems are addressed by CGP-UCB by choosing $\mathbf{K}_S = \mathbf{I}$ as the $m \times m$ identity matrix, and $\mathbf{K}_Z$

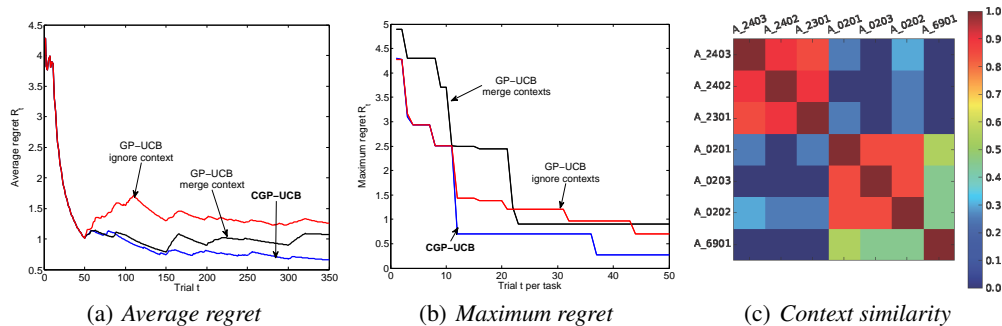

|(a) *Average regret*|(b) *Maximum regret*|(c) *Context similarity*|

Figure 2: CGP-UCB applied to the average (a) and maximum regret over all molecules (b) for three methods on MHC benchmark. (c) Context similarity using inter task predictions.

as the linear kernel on the features[2]. In this application, additive kernel combinations may be useful to model temporal dependencies of the overall click probabilities (e.g., during evening, users may or may not be more likely to click on an ad than during business hours).

**Learning to control complex systems.** Suppose we have a complex system and would like to achieve some desired behavior, for example robot walking [10]. In such a setting, we may wish to estimate a controller in a data-driven manner; however, we would also like to maximize the performance of the estimated controller, resulting in an exploration–exploitation tradeoff. In addition to controller parameters $\mathbf{s} \in S \subseteq \mathbb{R}^{d_S}$, the system may be exposed to changing (in an uncontrollable manner) environmental conditions, which are provided as context $\mathbf{z} \in Z \subseteq \mathbb{R}^{d_Z}$. The goal is thus to learn, which control parameters to apply in which conditions to maximize system performance. In this case, we may consider using a linear kernel $k_Z(\mathbf{z}, \mathbf{z}') = \mathbf{z}^T \mathbf{z}'$ to model the dependence of the performance on environmental features, and a squared exponential kernel $k_S(\mathbf{s}, \mathbf{s}')$ to model the smooth but nonlinear response of the system to the chosen control parameters. Theorems 1 and 2 bound $R_T = \mathcal{O}^*(\sqrt{T d_Z (\log T)^{d_S+1}})$. Additive kernel combinations may allow to model the fact that control in some contexts (environments) is inherently more difficult (or noisy).

**Multi-task experimental design.** Suppose we would like to perform a sequence of related experiments. In particular, in Section 6.1 we consider the case of vaccine design. The aim is to discover peptide sequences which bind to major histocompatibility complex molecules (MHC). MHC molecules present fragments of proteins from within the cell to T cells, resulting in healthy cells being left alone, while cells containing foreign proteins to be attacked by the immune system. Here, each experiment is associated with a set of features (encoding the MHC alleles), which are provided as context $\mathbf{z}$. The goal in each experiment is to choose a stimulus (the vaccine) $\mathbf{s} \in S$ that maximizes an observed response (binding affinity). In this case, we may consider using a finite inter-task covariance kernel $\mathbf{K}_Z$ with rank $m_Z$ to model the similarity of different experiments, and a Gaussian kernel $k_S(\mathbf{s}, \mathbf{s}')$ to model the smooth but nonlinear dependency of the stimulus response on the experimental parameters. Theorems 1 and 2 bound $R_T = \mathcal{O}^*(\sqrt{T m_Z (\log T)^{d_S+1}})$.

**Spatiotemporal monitoring with sensor networks.** Suppose we have deployed a network of sensors, which we wish to use to monitor the maximum temperature in a building. Due to battery limitations, we would like, at each timestep, to only activate few sensors. We can cast this problem in the contextual bandit setting, where time of day is considered as the context $\mathbf{z} \in Z$, and each action $\mathbf{s} \in S$ corresponds to picking a sensor. Due to the fact that the sun is moving relative to the building, the hottest point in the building changes depending on the time of the day, and we would like to learn which sensors to activate at which time of the day. In this problem, we would estimate a joint spatio-temporal covariance function (e.g., using the Matérn kernel), and use it for inference. We show experimental results for this problem in Section 6.2.

## 6 Experiments

In our two experimental case studies, we aim to study how much context information can help. We compare three methods: Ignoring (correlation between) contexts by running a separate instance of GP-UCB for every context (i.e., ignoring measurements from all but the current molecule or time);

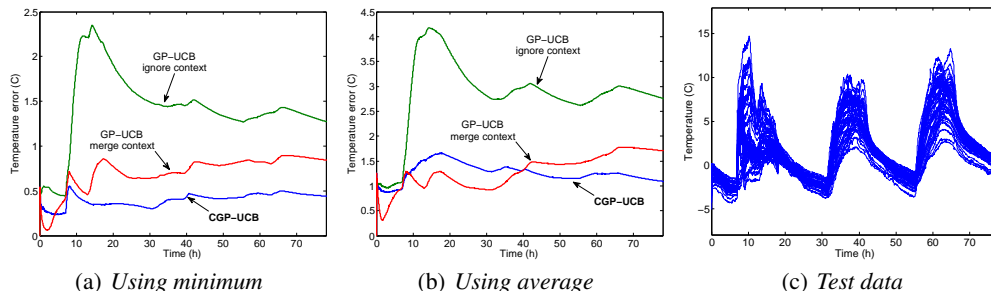

(a) *Using minimum*  (b) *Using average*  (c) *Test data*

Figure 3: CGP-UCB applied to temperature data from a network of 46 sensors at Intel Research Berkeley.

running a single instance of GP-UCB, merging together the context information (i.e., ignoring the molecule or time information); and running CGP-UCB, conditioning on measurements made at different contexts (MHC molecules considered / times of day) using the product kernel.

## 6.1 Multi-task Bayesian Optimization of MHC class-I binding affinity

We perform experiments in the multi-task vaccine design problem introduced in Section 5.3. In our experiments, we focus on a subset of MHC class I molecules that have affinity binding scores available. Each experimental design task corresponds to searching for maximally binding peptides, which is a vital step in the design of peptide-based vaccines. We use the data from [11], which is part of a benchmark set of MHC class I molecules [12]. The data contains binding affinities ($IC_{50}$ values), as well as features extracted from the peptides. Peptides with $IC_{50}$ values greater than 500 nM were considered non-binders, all others binders. We convert the $IC_{50}$ values into negative log scale, and normalize them so that 500nM corresponds to zero, i.e. $-\log_{10}(IC_{50}) + \log_{10}(500)$.

In total, we consider identifying peptides for seven different MHC molecules (i.e., seven related tasks = contexts). The context similarity was obtained using the hamming distance between amino acids in the binding pocket [11] (see Figure 2(c)), and we used the Gaussian kernel on the extracted features. We used a random subset of 1000 examples to estimate hyperparameters, and then considered each MHC allele in the order shown in Figure 2(c). For each MHC molecule, we ran CGP-UCB for 50 trials.

From Figure 2(a) we see that for the first three molecules (up to trial 150), which are strongly correlated, merging contexts and CGP-UCB perform similarly, and both perform better than ignoring observations from other MHC molecules previously considered. However, the fourth molecule (A 0201) has little correlation with the earlier ones, and hence simply merging contexts performs poorly. We also wish to study, how long it takes, in the worst-case over all seven molecules, to identify a peptide with binding affinity of desired strength. Therefore, in Figure 2(b), we plot, for each $t$ from 1 to 50, the largest (across the seven tasks) discrepancy between the maximum achievable affinity, and the best affinity score observed in the first $t$ trials. We find that by exploiting correlation among contexts, CGP-UCB outperforms the two baseline approaches.

## 6.2 Learning to Monitor Sensor Networks

We also apply CGP-UCB to the spatiotemporal monitoring problem described in Section 5. We use data from 46 sensors deployed at Intel Research, Berkeley. The data set contains 4 days of data, sampled at 5 minute intervals. We take the first 24 hours to fit (by maximizing the marginal likelihood) parameters of a spatio-temporal covariance function (we choose the Matérn kernel with $\nu = 2.5$). On the remaining 3 days of data (see Figure 3(c)), we then proceed by, at each time step, sequentially activating 5 sensors and reporting the regret of the average and maximum temperature measured (hereby the regret is the error in estimating the actual maximum temperature reported by any of the 46 sensors).

Figure 3(a) (using the maximum temperature among the 5 readings each time step) and 3(b) (using the average temperature) show the results of this experiment. Notice that ignoring contexts performs poorly. Merging contexts (single instance of context-free GP-UCB) performs best for the first few timesteps (since temperature is very similar, and the highest temperature sensor does not change). However, after running CGP-UCB for more than one day of data (i.e., until context reoccurs), it outperforms the other methods, since it is able to learn to query the maximum temperature sensors as a function of the time of the day.

# 7 Related Work

The use of upper confidence bounds to trade off exploration and exploitation has been introduced by [13], and studied thereafter [1, 14, 15, 16]. The approach for the classical $k$-armed bandit setting [17] has been generalized to more complex settings, such as infinite action sets and linear payoff functions [14, 18], Lipschitz continuous payoff functions [15] and locally-Lipschitz functions [19]. However, there is a strong tradeoff between strength of the assumptions and achievable regret bounds. For example, while $\mathcal{O}(d\sqrt{T \log T})$ can be achieved in the linear setting [14], if only Lipschitz continuity is assumed, regret bounds scale as $\Omega(T^{\frac{d+1}{d+2}})$ [15]. Srinivas et al [3] analyze the case where the payoff function is sampled from a GP, which encodes configurable assumptions. The present work builds on and strictly generalizes their approach. In fact, in the context free case, CGP-UCB is precisely the GP-UCB algorithm of [3]. The ability to incorporate contextual information, however, significantly expands the class of applications of GP-UCB. Besides handling context and bounding the stronger notion of contextual regret, in this paper we provide generic techniques for obtaining regret bounds for composite kernels. An alternative rule (in the context free setting) is the Expected Improvement algorithm [20], for which no bounds on the cumulative regret are known.

For contextual bandit problems, work has focused on the case of finitely many actions, where the goal is to obtain sublinear contextual regret against classes of functions mapping context to actions [1]. This setting resembles (multi-class) classification problems, and regret bounds can be given in terms of the VC dimension of the hypothesis space [2]. [6] present an approach, LinUCB, that assumes that payoffs for each action are linear combinations (with unknown coefficients) of context features. In [5], it is proven that a modified variant of LinUCB achieves sublinear contextual regret. Theirs is a special case of our setting (assuming a linear kernel for the contexts and diagonal kernel for the actions). Another related approach is taken by Slivkins [21], who presents several algorithms with sublinear contextual regret for the case of infinite actions and contexts, assuming Lipschitz continuity of the payoff function in the context-action space. In [22], this approach is generalized to select *sets* of actions, and applied to a problem of diverse retrieval in large document collections. However, in contrast to CGP-UCB, this approach does not enable stronger guarantees for smoother or more structured payoff functions.

The construction of composite kernels is common in the context of multitask learning with GPs [23, 24, 25]. Instead of considering a scalar GP with joint feature space $f : S \times Z \to \mathbb{R}$, they consider a multioutput GP $f_{vec} : S \to \mathbb{R}^Z$, and introduce output correlations as linear combinations of latent channels or convolutions of GPs [25]. Our results are complementary to this line of work, as we can make use of such kernel functions for "multi-task Bayesian optimization". Theorems 2 and 3 provide convenient ways for deriving regret bounds for such problems. There has been a significant amount of work on GP optimization and response surface methods [26]. For example, [27] consider sharing information across multiple sessions in a problem of parameter identification in animation design. We are not aware of theoretical convergence results in case of context information, and our Theorem 1 provides the first general approach to obtain rates.

# 8 Conclusions

We have described an algorithm, CGP-UCB, which addresses the exploration–exploitation tradeoff in a large class of contextual bandit problems, where the regularity of the payoff function defined over the action–context space is expressed in terms of a GP prior. As we discuss in Section 5, by considering various kernel functions on actions and contexts this approach allows to handle a variety of applications. We show that, similar as in the context free case studied by [3], the key quantity governing the regret is a mutual information between experiments performed by CGP-UCB and the GP prior (Theorem 1). In contrast to prior work, however, our approach bounds the much stronger notion of contextual regret (competing with the optimal mapping from contexts to actions). We prove that in many practical settings, as discussed in Section 5, the contextual regret is sublinear. In addition, Theorems 2 and 3 provide tools to construct bounds on this information theoretic quantity given corresponding bounds on the context and actions. We also demonstrate the effectiveness of CGP-UCB on two applications: computational vaccine design and sensor network management. In both applications, we show that utilizing context information in the joint covariance function reduces regret in comparison to ignoring or naively using the context.

**Acknowledgments** The authors wish to thank Christian Widmer for providing the MHC data, as well as Daniel Golovin and Aleksandrs Slivkins for helpful discussions. This research was partially supported by ONR grant N00014-09-1-1044, NSF grants CNS-0932392, IIS-0953413, DARPA MSEE grant FA8650-11-1-7156 and SNF grant 200021_137971.

## Footnotes

[1]We will also consider the case where $f$ has low norm in the RKHS associated with the covariance $k$.

[2][6] also propose a more complex hybrid model that uses features shared between the actions. This model is also captured in our framework by adding a second kernel function, which composes a low-rank (instead of $\mathbf{I}$) matrix with the linear kernel.

# References

[1] Peter Auer. Using confidence bounds for exploitation-exploration trade-offs. *JMLR*, 3, 2002.

[2] John Langford and Tong Zhang. The epoch-greedy algorithm for contextual multi-armed bandits. In *NIPS*, 2008.

[3] N. Srinivas, A. Krause, S. Kakade, and M. Seeger. Gaussian process optimization in the bandit setting: No regret and experimental design. In *ICML*, 2010.

[4] C. E. Rasmussen and C. K. I. Williams. *Gaussian Processes for Machine Learning*. MIT Press, 2006.

[5] Wei Chu, Lihong Li, Lev Reyzin, , and Robert E. Schapire. Contextual bandits with linear payoff functions. In *AISTATS*, 2011.

[6] Lihong Li, Wei Chu, John Langford, and Robert E. Schapire. A contextual-bandit approach to personalized news article recommendation. In *WWW*, 2010.

[7] T. M. Cover and J. A. Thomas. *Elements of Information Theory*. Wiley Interscience, 1991.

[8] G. Wahba. *Spline Models for Observational Data*. SIAM, 1990.

[9] Naoki Abe, Alan W. Biermann, and Philip M. Long. Reinforcement learning with immediate rewards and linear hypotheses. *Algorithmica*, 37(4):263–293, 2003.

[10] D. Lizotte, T. Wang, M. Bowling, and D. Schuurmans. Automatic gait optimization with Gaussian process regression. In *IJCAI*, pages 944–949, 2007.

[11] C. Widmer, N. Toussaint, Y. Altun, and G. Rätsch. Inferring latent task structure for multitask learning by multiple kernel learning. *BMC Bioinformatics*, 11(Suppl 8:S5), 2010.

[12] B. Peters et. al. A community resource benchmarking predictions of peptide binding to mhc-i molecules. *PLoS Computational Biology*, 2(6):e65, 2006.

[13] T. L. Lai and H. Robbins. Asymptotically efficient adaptive allocation rules. *Adv. Appl. Math.*, 6:4, 1985.

[14] V. Dani, T. P. Hayes, and S. Kakade. The price of bandit information for online optimization. In *NIPS*, 2007.

[15] R. Kleinberg, A. Slivkins, and E. Upfal. Multi-armed bandits in metric spaces. In *STOC*, pages 681–690, 2008.

[16] L. Kocsis and C. Szepesvári. Bandit based monte-carlo planning. In *ECML*, 2006.

[17] P. Auer, N. Cesa-Bianchi, and P. Fischer. Finite-time analysis of the multiarmed bandit problem. *Mach. Learn.*, 47(2-3):235–256, 2002.

[18] V. Dani, T. P. Hayes, and S. M. Kakade. Stochastic linear optimization under bandit feedback. In *COLT*, 2008.

[19] S. Bubeck, R. Munos, G. Stoltz, and C. Szepesvári. Online optimization in X-armed bandits. In *NIPS*, 2008.

[20] S. Grünewälder, J-Y. Audibert, M. Opper, and J. Shawe-Taylor. Regret bounds for gaussian process bandit problems. In *AISTATS*, 2010.

[21] Aleksandrs Slivkins. Contextual bandits with similarity information. Technical Report 0907.3986, arXiv, 2009.

[22] Aleksandrs Slivkins, Filip Radlinski, and Sreenivas Gollapudi. Learning optimally diverse rankings over large document collections. In *ICML*, 2010.

[23] Kai Yu, Volker Tresp, and Anton Schwaighofer. Learning gaussian processes from multiple tasks. In *ICML*, 2005.

[24] Edwin V. Bonilla, Kian Ming A. Chai, and Christopher K. I. Williams. Multi-task gaussian process prediction. In *NIPS*, 2008.

[25] Mauricio A. Álvarez, David Luengo, Michalis K. Titsias, and Neil D. Lawrence. Efficient multioutput gaussian processes through variational inducing kernels. In *AISTATS*, 2010.

[26] E. Brochu, M. Cora, and N. de Freitas. A tutorial on Bayesian optimization of expensive cost functions, with application to active user modeling and hierarchical reinforcement learning. In *TR-2009-23, UBC*, 2009.

[27] Eric Brochu, Tyson Brochu, and Nando de Freitas. A bayesian interactive optimization approach to procedural animation design. In *Eurographics*, 2010.

